# IR-CM: The Fast and General-purpose Image Restoration Method Based on Consistency Model

**Xiaoxuan Gong**
School of Artificial Intelligence and Automation, HUST
Huazhong University of Science and Technology
Wuhan, China
`gongxiaoxuan286@gmail.com`

**Jie Ma***
School of Artificial Intelligence and Automation, HUST
Huazhong University of Science and Technology
Wuhan, China
`majie@hust.edu.cn`

## Abstract

This paper proposes a fast and general-purpose image restoration method. The key idea is to achieve few-step or even one-step inference by conducting consistency distilling or training on a specific mean-reverting stochastic differential equations. Furthermore, based on this, we propose a novel linear-nonlinear decoupling training strategy, significantly enhancing training effectiveness and surpassing consistency distillation on inference performance. This allows our method to be independent of any pre-trained checkpoint, enabling it to serve as an effective standalone image-to-image transformation model. Finally, to avoid trivial solutions and stabilize model training, we introduce a simple origin-guided loss. To validate the effectiveness of our proposed method, we conducted experiments on tasks including image deraining, denoising, deblurring, and low-light image enhancement. The experiments show that our method achieves highly competitive results with only one-step inference. And with just two-step inference, it can achieve state-of-the-art performance in low-light image enhancement. Furthermore, a number of ablation experiments demonstrate the effectiveness of the proposed training strategy. our code is available at `https://github.com/XiaoxuanGong/IR-CM`.

## 1   Introduction

Image restoration is a classic problem in the field of computer vision. It aims to transform low-quality or noisy images into their corresponding high-quality or noise-free counterparts. In many industrial applications (such as autonomous driving), there are various complex types of image degradation, including rain and fog interference, glare interference, low-light conditions, and motion blur etc. This necessitates high generality in image restoration algorithms. Moreover, there is often a high demand for real-time performance in practical application scenarios, posing even greater challenges to the design of algorithms and models.

Common image restoration tasks include image deraining[1, 2, 3, 4, 5, 6], denoising[7, 8, 9, 10, 11], and deblurring[12, 13, 14, 15] etc. Due to the demands of industrial applications, research on low-light image enhancement[16, 17, 18, 19, 20, 21] is also gradually increasing. However, these methods are often heuristic and difficult to apply to general tasks because they typically require domain-specific prior knowledge for training. A more general approach is the recently proposed IR-SDE[22], which

does not rely on any prior knowledge. It only requires pairs of high-quality(HQ)/low-quality(LQ) images for training and achieves good results across multiple image restoration tasks. However, like most diffusion-based methods, it requires multi-step sampling for inference, making it difficult to meet the real-time requirements of practical applications.

In recent years, diffusion models have achieved remarkable results in both unconditional image generation[23, 24, 25, 26] and conditional image generation tasks[27, 28, 20, 29, 22, 30, 21, 31, 32]. In a more general description, diffusion models can be described using stochastic differential equations (SDEs)[33, 34]. It achieves this by converting the original data distribution into a fixed Gaussian distribution, then learning the corresponding distribution gradients (score) through a network, and finally gradually reconstructing the original image using SDE solvers or ODE solvers. Most SDE-based methods can generate high-quality samples, however they often require many steps of sampling to accomplish this. Despite the existence of many methods for accelerating sampling[25, 35, 36], it remains challenging to meet the real-time requirements in industrial applications. Recently, the consistency model[37] has been proposed. It aims to map any point on the ODE trajectory of the SDE-based model to its origin. Once trained, the model can achieve one-step inference with some decrease in model performance. The training of consistency models is divided into two approaches: consistency distillation(CD) and consistency training(CT). The former involves distilling training based on the ODE trajectories of a pre-trained SDE model. The latter, while not requiring a pre-trained SDE model and can be considered as an independent generative model, typically exhibits lower performance compared to the former[37]. This means that training consistency models often rely on pre-trained models, which inevitably leads to performance degradation.

The purpose of this paper is to design a universal image restoration model with fast inference. The proposed model, named IR-CM (*Image Restoration Consistency Model*), achieves one-step or few-step fast inference through consistency model training. Due to its versatility and flexibility in different image transformation tasks, the IR-SDE serves as the foundation for our method. IR-CM can be trained using consistency distillation (CD) on a pre-trained model, but at the cost of a slight decrease in performance. Therefore, we focus more on consistency training (CT), enabling IR-CM to become an independent image restoration model without relying on any pre-trained checkpoint. Furthermore, to improve the effectiveness of CT, we propose a novel linear-nonlinear decoupling training strategy and a novel origin-estimated consistency function, these allows the model's performance to reach or even surpass that of the original model. Finally, we propose a simple origin-guided loss to stabilize the training process. In summary, the main contributions of our method are as follows:

- We propose a universal and fast image restoration method that can obtain high-quality images with one step or few-step sampling. For different tasks, training only requires dataset replacement, without the need for any additional prior knowledge.
- We have introduced a novel linear-nonlinear decoupling training strategy, enabling our method to achieve even surpass the performance of the original model without relying on any pre-trained checkpoint.
- We propose a novel origin-estimated consistency function, which enables our model to have a more stable initial state and a smaller solution space, and a simple origin-guided loss to stabilize the training process. This makes our method more robust. The ablative experiments demonstrated its effectiveness.
- Our method achieves highly competitive performance in multiple tasks (including image deraining, denoising, deblurring, and low-light image enhancement) with one-step inference. With two-step inference, our method achieves state-of-the-art performance in low-light image enhancement task.

## 2 Preliminaries

### 2.1 Mean-reverting stochastic differential equation

Our method requires a SDE-based model as the base model for consistency training. Specifically, we choose IR-SDE[22] as the base SDE model due to its excellent generality and applicability. Its forward process involves gradually transforming high-quality images into corresponding low-quality versions with fixed-variance Gaussian noise. It can be represented as follows:

$$dx = \theta_t(\mu - x)dx + \sigma_t dw, \tag{1}$$

where $\mu \in \mathbb{R}^d$ typically represents the low-quality image, and $x(0) \in \mathbb{R}^d$ represents its corresponding high-quality version, $\theta_t$, $\sigma_t$ are time-dependent positive parameters, and they satisfy $\sigma_t^2/\theta_t = 2\lambda^2$ for all times $t$ with positive constant $\lambda$, and $dw$ represents Brownian motion. In [22], it has been proven that at each time t, the marginal probability distribution of x can be represented as follows:

$$p_t(x) = \mathcal{N}(x(t) \mid m_t, v_t), \tag{2}$$
$$m_t := \mu + (x(0) - \mu)e^{-\bar{\theta}_t},$$
$$v_t := \lambda^2(1 - e^{-2\bar{\theta}_t}),$$

where $\bar{\theta}_t = \int_0^t \theta_k dk$. As $t$ increases gradually, $m_t$ approaches $\mu$ and $v_t$ approaches $\lambda^2$. Thus, $x(0)$ (the high-quality image) will gradually transform into its corresponding low-quality version $\mu$ accompanied by Gaussian noise with variance $\lambda^2$.

With the above conclusion, we can naturally sample $x(t)$ by $x(t) = m_t + \sqrt{v_t}\epsilon_t$. Then we can train a network to estimate the noise $\epsilon_t$. In the inference phase, the reverse-time process of the IR-SDE can be represented as follows[34]:

$$dx = \left[\theta_t(\mu - x) - \frac{1}{2}\sigma_t^2 \nabla_x \log p_t(x)\right] dt, \tag{3}$$
$$\nabla_x \log p_t(x) = -\frac{x(t) - m_t}{v_t}$$
$$= -\frac{\hat{\epsilon}(x, \mu, t)}{\sqrt{v_t}},$$

where $\nabla_x \log p_t(x)$ is called score function and $\hat{\epsilon}(x, \mu, t)$ is the noise estimated by network. Then, similar to other SDE-based models, we can adopt a SDE-solver (or ODE-solver) to reverse the process to restore the low-quality image $x(t)$ back to the high-quality version $x(0)$ progressively.

## 2.2 Consistency model

For a solution trajectory $\{x_t\}_{t \in [\eta, T]}$ of any PF-ODE such as (3), a consistency function can be defined as $f(x_t, t) \equiv x_\eta$, where $\eta$ is a small positive number. This means that when sampling any pair $(x_t, t)$ on the trajectory of the PF-ODE, the output of the consistency function is always the initial point $x_\eta$ of the trajectory. This property is referred to as self-consistency[37]. The example of the consistency function in [37] is as follows:

$$f_\phi(x_t, t) = c_{skip}(t)x_t + c_{out}(t)F_\phi(x_t, t), \tag{4}$$

where $c_{skip}(t), c_{out}(t)$ are differentiable functions, and they satisfy $c_{skip}(\eta) = 1, c_{out}(\eta) = 0$, $F_\phi(x_t, t)$ is a trainable network initialized by a pre-trained noise estimation model. Once training is complete, we only need to input $x_T$ and apply $f_\phi(x_T, T) = x_\eta$ to obtain high-quality sample in one step. There are two training methods for consistency models:

**Consistency Distillation (CD)** For a discrete time sequence $t_1 = \eta < t_2 < ... < t_N = T$, given an arbitrary point $(x_{t_{n+1}}, t_{n+1})$ on PF-ODE trajectory, we can estimate the $x_{t_n}$ by following formula:

$$\hat{x}_{t_n}^\varphi = x_{t_{n+1}} + (t_n - t_{n+1})\Phi(x_{t_{n+1}}, t_{n+1}; \varphi), \tag{5}$$

where $\Phi(\cdot)$ represents the update function of a one step ODE solver applied to the PF-ODE, and $\varphi$ is the weights of a pre-trained score matching network. Then the CD loss can represent as follow:

$$\mathcal{L}_{CD}(\phi, \phi^-; \varphi) := \mathbb{E}\left[\lambda(t_n)d(f_\phi(x_{t_{n+1}}, t_{n+1}), f_{\phi^-}(\hat{x}_{t_n}^\varphi, t_n))\right], \tag{6}$$

where $\phi^-$ represents the exponential moving average (EMA) version of training weights $\phi$, it is frozen during backward, and $\lambda(\cdot)$ is a positive weighting function, $d(\cdot)$ denotes a distance function, such as the L1 or L2 distance. This approach essentially involves distillation training on the pre-trained score matching network, hence referred to as consistency distillation.

**Consistency Training (CT)** Unlike CD, the CT does not rely on a pre-trained score matching network and can independently train any SDE-based model. The CT loss is represented as follow:

$$\mathcal{L}_{CT}(\phi, \phi^-) := \mathbb{E}\left[\lambda(t_n)d(f_\phi(x_{t_{n+1}}, t_{n+1}), f_{\phi^-}(x_{t_n}, t_n))\right], \tag{7}$$

Here $x_{t_n}$ and $x_{t_{n+1}}$ are both sampled from the forward process of SDE model.

# 3 Method

The key idea of our method lies in employing our proposed two-stage training strategy and the origin-estimated consistency function to conduct consistency training on the IR-SDE model. We thus refer to it as an *Image Restoration Consistency Model* (IR-CM).We begin by describing the novel origin-estimated consistency function, followed by an explanation of the proposed two-stage training strategy and the origin-guided loss. The overall architecture is depicted in Fig 1.

## 3.1 Origin-estimated consistency function

For any SDE-based model, each step of the reverse process is essentially a prediction of the PF-ODE solution[34]. In other words, any point on the PF-ODE trajectory actually contains information about the origin. Based on this, according to equations (2) and (3), the origin prediction function at each point on the PF-ODE trajectory of the IR-SDE is as follows:

$$\hat{x}(0) = \mu(1 - e^{\bar{\theta}_t}) + e^{\bar{\theta}_t}\left[x(t) - \sqrt{v_t}\hat{\epsilon}_\phi(x, \mu, t)\right]. \tag{8}$$

During the consistency training phase(both for CD and CT), differ from the formula (4) proposed in [37], we train using the following origin-estimated consistency function (OECF):

$$f_\phi(x_t, t) = c_{skip}(t)x(t) + c_{out}(t)\hat{x}_0(x, t; \phi), \tag{9}$$

where the $\hat{x}_0(x, t; \phi)$ is obtained by (8). Suppose the pre-trained score model matches the ground truth, i.e., $\forall t \in [\eta, T] : \hat{\epsilon}_\phi(x, \mu, t) = \epsilon(t) + o(\Delta t)$ and $c_{skip}(T) = 0, c_{out}(T) = 1$. For formulas (4) and (9), $c_{skip}(t)$, $x(t)$, and $c_{out}(t)$ are all same and constant at each time t. Thus, for simplicity in analysis, we specifically consider the moment $t = T$. Then substituting equation (2) into equations (4) and (9) respectively, we obtain:

$$f_\phi(x_T, T) = x(0) + \frac{x(T) - m_T - \sqrt{v_T}x(0)}{\sqrt{v_T}} + o(\Delta t), \tag{10}$$

$$f_{\phi OECF}(x_T, T) = x(0) - e^{\bar{\theta}_T}v_T o(\Delta t). \tag{11}$$

From equation (10), it can be observed that if equation (4) is chosen as the consistency function, the initial solution will have a significant fixed error. In contrast, using the OECF, the initial solution will only have a time-dependent higher-order infinitesimal error. Therefore, for a pre-trained score matching model, the OECF offers a more stable initial state and hence a smaller solution space for all time $t$.

Intuitively, OECF can effectively enhance the training performance of CD. In practice, we have found that for CT, OECF also exhibits significant performance improvements compared to equation (4). The relevant ablative experiment results will be presented in Chapter 4. Note that, unlike CD, the pre-trained model in CT is only used to initialize the training weights and is not involved in any training process. In practice, training with randomly initialized weights is also feasible, albeit usually resulting in slightly longer convergence times.

## 3.2 Origin-guided loss

In general training, we randomly sample $(x_t, t)$ and $(x_{t+1}, t + 1)$ using equation (2), and then simply apply (6) for CD or (7) for CT, as illustrated in the lower half of Figure 1. However, we have empirically found that its performance is not ideal, and occasionally, pattern collapse occurs. Upon further analysis, we discovered that this often occurs when $t$ is not sufficiently small during the random sampling in the early stages of training. This also leads to the emergence of mediocre solutions. Below we provide a simple theoretical proof.

**Theorem 1.** Let $c_{out}(t)$ be monotonic differentiable and satisfy $c_{out}(\eta) = 0, c_{out}(T) = 1$, consider (9) as consistency function. When $t > \eta$, for any $y \in \mathbb{R}^d$, there always exists a $\hat{\epsilon}_\phi(x_t, \mu, t)$ such that $f_\phi(x_t, t) = y$.

*Proof.* Recall (9), there is

$$y = c_{skip}(t)x(t) + c_{out}(t)\hat{x}_0(x, t; \phi), \tag{12}$$

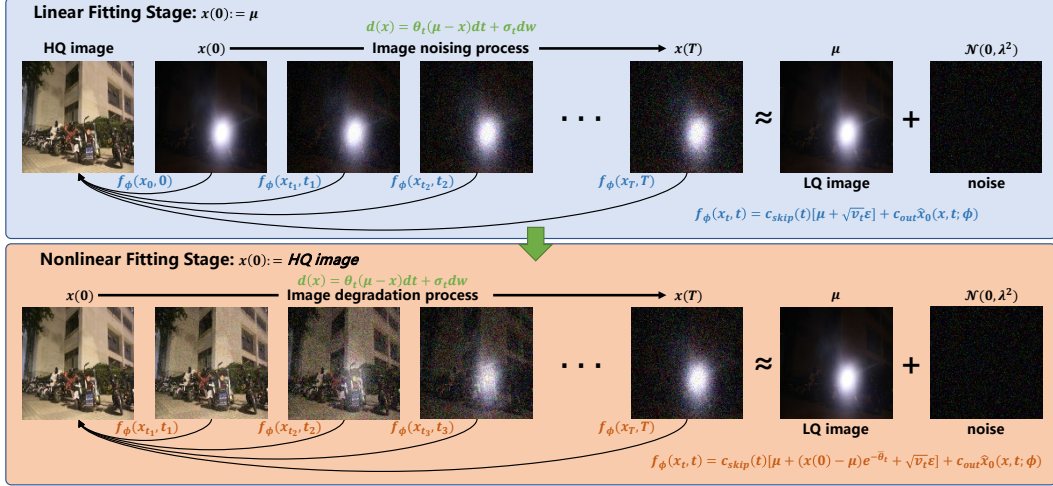

Figure 1: Two-stage training strategy, where $\mu$ is set as the low-quality image to be restored. In Stage One, by setting $x(0) = \mu$, the forward SDE (1) actually degrades into a simple noise addition process, upon which consistency training is conducted on $x(t)$ with different levels of noise. In Stage Two, $x(0)$ is set as the high-quality image and gradually transformed into the corresponding low-quality image with a fixed level of noise by (1). Consistency training is then performed on each intermediate state $x(t)$.

then substitute (8) into (12), we obtain:

$$\hat{\epsilon}_\phi(x_t, \mu, t) = e^{-\hat{\theta}_t} \frac{y - c_{skip}(t)x(t)}{c_{out}(t)v_t} + \frac{\mu}{v_t}(e^{-\hat{\theta}_t} - 1) + \frac{x(t)}{v_t}, \tag{13}$$

Note that when $t > \eta$, both $v_t$ and $\hat{\theta}_t$ are greater than 0, so the right-hand side of equation (13) is non-singular. This implies that under loss functions (6) and (7), the consistency function admits arbitrary non-singular solutions, leading to pattern collapse. In other hand, (6) and (7) only emphasize the self-consistency between any two points on the PF-ODE trajectory, leading to a lack of determinism in the training process. This uncertainty indirectly contributes to slower convergence of the model.

To address this issue, we additionally introduce the following origin-guided(OG) loss to stabilize the training process.

$$\mathcal{L}_{OG} = \mathbb{E}\left[\|f_\phi(x_{t+1}, t+1) - x(0)\|_1\right]$$
$$+ \lambda_{perc}\mathbb{E}\left[\|\Phi(f_\phi(x_{t+1}, t+1)) - \Phi(x(0))\|^2\right], \tag{14}$$

where $\lambda_{perc}$ is a positive constant, and $\Phi(\cdot)$ represents a VGG16[38] feature extractor from 2nd and 5th pooling layers. This is equivalent to performing a consistency computation with the origin after each random sampling, hence referred to as the origin-guided loss. This effectively avoids the emergence of mediocre solutions and adds some certain determinism to the training process, resulting in faster convergence. Then the final loss function is represented as follows:

$$\mathcal{L}_{full} = \mathcal{L}_{CD/CT} + \lambda_{OG}\mathcal{L}_{OG}, \tag{15}$$

where $\lambda_{OG}$ is a positive constant. The selection of $\lambda_{OG}$ will be discussed in the ablation experiment section of Chapter 4.

### 3.3 Linear-nonlinear decoupling training strategy

With the foundation laid in Sections 3.1 and 3.2, we can naturally train IR-CM by a regular CD or CT process, as illustrated in the lower half of Figure 1. Despite this method offers convenience in training IR-CM, we have empirically found that its performance is not optimal in practice. Let us recall the SDE (2) and OECF (9), we obtain:

$$f_\phi(x_t, t) = \underbrace{c_{skip}(t)\mu}_{\text{linear state } f_1} + \underbrace{c_{skip}(t)\left[(x(0) - \mu)e^{-\bar{\theta}_t}\right]}_{\text{nonlinear intermediate state } f_2} + \underbrace{c_{skip}(t)\sqrt{v_t}\epsilon}_{\text{noise } f_3} + c_{out}(t)\hat{x}_0(x, t; \phi). \tag{16}$$

**Algorithm 1:** Linear-fitting stage

**Input** dataset $\mathcal{D}$, model parameter $\phi$, OECF $f_\phi(\cdot,\cdot)$, learning rate $\xi$, OG Weight $\lambda_{OG}$;
$\phi^- \leftarrow \phi$;
**while** *not convergence* **do**
> Sample $(x_{LQ}, y_{HQ}) \sim \mathcal{D}$ and $n \sim \mathcal{U}[1, N-1]$;
> $\mu \leftarrow x_{LQ}, x(0) \leftarrow x_{LQ}$;
> Sample $x_{t_n} \sim \mathcal{N}(m_{t_n}, v_{t_n})$ and $x_{t_{n+1}} \sim \mathcal{N}(m_{t_{n+1}}, v_{t_{n+1}})$;
> $\mathcal{L}_{full}(\phi, \phi^-) \leftarrow$
> $\mathcal{L}_{CT}(f_\phi(x_{t_n}, t_n), f_\phi(x_{t_{n+1}}, t_{n+1})) +$
> $\lambda_{OG}\mathcal{L}_{OG}(f_\phi(x_{t_{n+1}}, t_{n+1}), y_{HQ})$;
> $\phi \leftarrow \phi - \xi\nabla\mathcal{L}_{full}(\phi, \phi^-)$
> $\phi \leftarrow \text{stopgrad}(\phi)$

**end**

**Algorithm 2:** NonLinear-fitting stage

**Input** dataset $\mathcal{D}$, model parameter $\phi$, OECF $f_\phi(\cdot,\cdot)$, learning rate $\xi$, OG Weight $\lambda_{OG}$;
$\phi^- \leftarrow \phi$;
**while** *not convergence* **do**
> Sample $(x_{LQ}, y_{HQ}) \sim \mathcal{D}$ and $n \sim \mathcal{U}[1, N-1]$;
> $\mu \leftarrow x_{LQ}, x(0) \leftarrow y_{HQ}$;
> Sample $x_{t_n} \sim \mathcal{N}(m_{t_n}, v_{t_n})$ and $x_{t_{n+1}} \sim \mathcal{N}(m_{t_{n+1}}, v_{t_{n+1}})$;
> $\mathcal{L}_{full}(\phi, \phi^-) \leftarrow$
> $\mathcal{L}_{CT}(f_\phi(x_{t_n}, t_n), f_\phi(x_{t_{n+1}}, t_{n+1})) +$
> $\lambda_{OG}\mathcal{L}_{OG}(f_\phi(x_{t_{n+1}}, t_{n+1}), x(0))$;
> $\phi \leftarrow \phi - \xi\nabla\mathcal{L}_{full}(\phi, \phi^-)$
> $\phi \leftarrow \text{stopgrad}(\phi)$

**end**

In general training, the model attempts to simultaneously fit the variations of both linear part $f_1$ and nonlinear part $f_2$. Obviously, this is more challenging than fitting $f_1$ alone. In practice, the model's performance at larger values of $t$ is more crucial during training, because any intermediate state for $x(t), \forall t \in [\eta, T)$ are unknown during inference, thus we can only set $t = T$ for inference. And note that there is $\lim_{t\to T} f_2 \approx 0$, therefore, the influence of $f_2$ is negligible when $t = T$. Based on the analysis above, we can then eliminate $f_2$ for all $t$ by simply setting $x(0)$ as $\mu$. By doing so, the original image degradation process is transformed into a purely image noising process (as shown in the upper half of Figure 1). Since only the linear part $f_1$ and noise $f_3$ are being fitted, the model will achieve better performance over the entire PF-ODE trajectory, leading to improved inference performance as well. We refer to this training as the linear-fitting stage and the corresponding pseudocode is shown in Algorithm 1.

Despite the Linear-fitting stage achieves good performance for one-step inference after training, we can still further improve the performance by employing appropriate multi-step sampling. However, the model trained in the Linear-fitting stage cannot perform multi-step sampling inference because the outputed HQ image is not the origin $x(0) = \mu$ of PF-ODE trajectory, making it unable to estimate any intermediate states $x_t$ after one inference. Therefore, after the Linear-fitting stage, we set $x(0)$ to be the HQ image instead of $\mu$ and fine-tune the model (Algorithm 2). Since in the previous stage, the model has already fitted $f_1$ and $f_3$, this stage mainly focuses on fitting the non-linear part $f_2$. Therefore, this stage is naturally referred to as the non-Linear-fitting stage. After training in the non-linear stage, the model's performance at $t = T$ is almost unchanged, meaning there is no change in the performance of one-step inference. But the model will be able to perform multi-step sampling inference (Algorithm 3 in appendix C). In practice, even just two-step sampling inference brings a noticeable performance improvement. Relevant comparative experiments and ablative experiments will be presented in Chapter 4.

## 4 Experiments

### 4.1 Comparative Experiment

In this section, we will validate the effectiveness of our proposed method on five tasks: image de-raining, image denoising, image deblurring, low-light image enhancement, and nighttime glare removal. The implementation details are described in Appendix A. Specifically, we will compare our method with some of the state-of-the-art methods on PSNR, SSIM[39], LPIPS[40] metrics and NFE. The NFE (Number of Function Evaluations) refers to the number of function evaluations required to generate an image or data. In other words, it is the number of evaluations needed at each step of the diffusion process. Notably, like other SDE-based models, we prioritize perceptual scores LPIPS over distortion scores PSNR and SSIM. And our metric settings are same as other mentioned baseline methods. For PSNR metric, we perform the calculation in the luminance space (Y channel). For SSIM metric, it refer to [39], and for LPIPS metric, it refer to [40]. All comparison experiments were

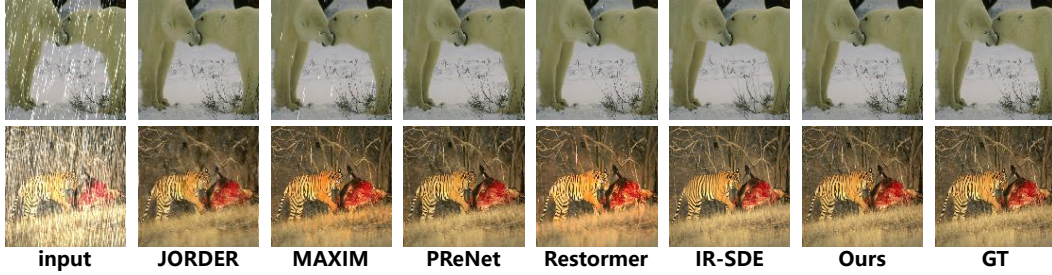

| input | JORDER | MAXIM | PReNet | Restormer | IR-SDE | Ours | GT |

Figure 2: Qualitative comparison results on R100L dataset (upper row) and R100H dataset (bottom row). More visual results are available in appendix D.

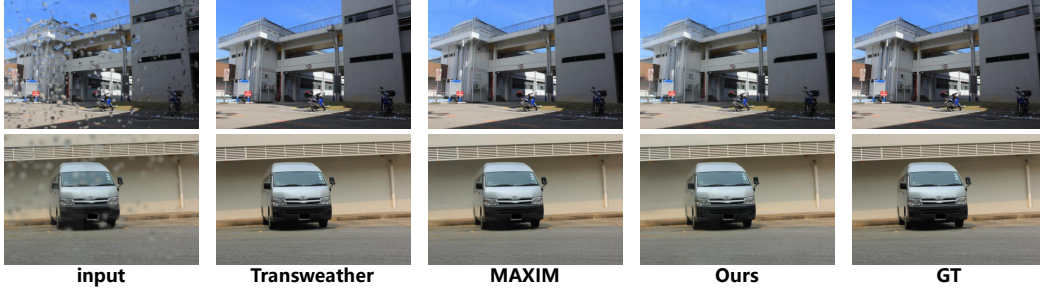

| input | Transweather | MAXIM | Ours | GT |

Figure 3: Qualitative comparison results on Raindrop dataset.

| Methods | Metrics | | | |
|---|---|---|---|---|
| Methods / Metrics | PSNR↑ | SSIM↑ | LPIPS↓ | NFE↓ |
| JORDER[3] | 26.25 | 0.83 | 0.197 | 1 |
| MAXIM[4] | 30.81 | 0.90 | 0.133 | 1 |
| PReNet[1] | 29.46 | 0.90 | 0.128 | 6 |
| Restormer[5] | 31.46 | 0.90 | 0.086 | 1 |
| IR-SDE[22] | **31.65** | 0.90 | 0.047 | 100 |
| CNN-baseline | 29.12 | 0.88 | 0.153 | 1 |
| IR-CM-CD (ours) | 29.75 | 0.88 | 0.064 | 1 |
| IR-CM-CT (ours) | 30.47 | 0.92 | 0.016 | 1 |
| IR-CM-CT (ours) | 30.71 | **0.92** | **0.015** | 2 |

Table 1: Quantitative comparison results on R100H dataset. The optimal results are indicated by **bold underlining**.

| Methods | Metrics | | | |
|---|---|---|---|---|
| Methods / Metrics | PSNR↑ | SSIM↑ | LPIPS↓ | NFE↓ |
| JORDER[3] | 36.61 | 0.97 | 0.028 | 1 |
| MAXIM[4] | 38.06 | 0.98 | 0.048 | 1 |
| PReNet[1] | 37.48 | 0.98 | 0.020 | 6 |
| Restormer[5] | **38.99** | 0.98 | 0.013 | 1 |
| IR-SDE[22] | 38.30 | 0.98 | 0.014 | 100 |
| CNN-baseline | 33.17 | 0.96 | 0.068 | 1 |
| IR-CM-CD (ours) | 34.21 | 0.96 | 0.035 | 1 |
| IR-CM-CT (ours) | 36.18 | 0.98 | 0.009 | 1 |
| IR-CM-CT (ours) | 37.06 | 0.98 | **0.005** | 2 |

Table 2: Quantitative comparison results on R100L dataset. The optimal results are indicated by **bold underlining**.

conducted using the original resolution of each dataset (for ease of presentation, the image sizes were adjusted in Figures 2–5).

### 4.1.1 Image deraining

We validate the effectiveness of the proposed IR-CM based on two datasets: R100L and R100H[41]. A total of 2000 images are used for training, while 200 images are reserved for testing. Our method is qualitatively and quantitatively compared with some of the state-of-the-art image deraining methods including IR-SDE[22], JORDER[3], Restormer[5], PReNet[1], and MAXIM[4]. The comparison results are shown in Table 1, Table 2 and Figure 2. More visual results are available in appendix D. To further validate the effectiveness of our method in real-world rainy scenarios, we conducted comparative experiments on the Raindrop[42] dataset containing 1119 pairs of real-world rainy/non-rainy images. The results are shown in Table 3 and Figure 3.

The IR-CM model we proposed surpasses the baseline IR-SDE model and achieves optimal performance on both SSIM and LPIPS metrics in scenarios of either one-step or two-step reasoning. Furthermore, the comparison results with the CNN-baseline demonstrate that our approach improves network performance while ensuring real-time capabilities. Note that our IR-CM is based on IR-SDE for CD or CT, thus IR-CM-CD represents consistency distillation based on pre-trained IR-SDE model as teacher model. And IR-CM-CT only initializes the model using pre-trained IR-SDE checkpoint of faster convergence and does not use any teacher model during the CT process. Of course, random

| methods / metrics | PSNR↑ | SSIM↑ | LPIPS↓ | NFE↓ |
|---|---|---|---|---|
| MAXIM(2022) | 31.87 | 0.935 | 0.079 | 1 |
| Transweather(2021) | **34.55** | **0.950** | 0.051 | 1 |
| Refusion(2023) | 32.61 | 0.938 | 0.048 | 100 |
| IR-CM(1-step)(ours) | 32.06 | 0.934 | 0.043 | 1 |
| IR-CM(2-step)(ours) | 32.89 | 0.936 | **0.041** | 2 |

Table 3: Quantitative comparison with some of image deraining methods on Raindrop dataset. The optimal results are indicated by **bold underlining**.

| Methods | Metrics | | | |
|---|---|---|---|---|
| Methods / Metrics | PSNR↑ | SSIM↑ | LPIPS↓ | NFE↓ |
| DeepDeblur[43] | 29.08 | 0.91 | 0.135 | 1 |
| DBGAN[15] | 31.18 | 0.92 | 0.112 | 1 |
| DeblurGAN-v2[14] | 29.55 | 0.93 | 0.117 | 1 |
| MAXIM[4] | 32.86 | 0.94 | 0.089 | 1 |
| IR-SDE[22] | 30.70 | 0.90 | 0.064 | 100 |
| DiffIR[44] | **33.20** | **0.963** | - | 4 |
| IR-CM-CD (ours) | 28.96 | 0.90 | 0.089 | 1 |
| IR-CM-CT (ours) | 29.72 | 0.95 | 0.013 | 1 |
| IR-CM-CT (ours) | 29.87 | 0.95 | **0.012** | 2 |

Table 4: Quantitative comparison results on Go-Pro dataset. The optimal results are indicated by **bold underlining**.

| Methods | Metrics | | | |
|---|---|---|---|---|
| Methods / Metrics | PSNR↑ | SSIM↑ | LPIPS↓ | NFE↓ |
| DnCNN[7] | 31.52 | 0.87 | 0.101 | 1 |
| FFDNet[8] | 32.36 | **0.89** | 0.103 | 1 |
| IR-SDE[22] | 29.48 | 0.81 | 0.071 | 100 |
| Denoising-SDE[22] | 28.98 | 0.75 | 0.088 | 22 |
| Denoising-ODE[22] | **32.39** | 0.88 | 0.055 | 22 |
| CNN-baseline | 25.02 | 0.78 | 0.102 | 1 |
| IR-CM-CD (ours) | 24.98 | 0.78 | 0.085 | 1 |
| IR-CM-CT (ours) | 25.61 | 0.82 | 0.048 | 1 |
| IR-CM-CT (ours) | 27.56 | 0.88 | **0.027** | 2 |

Table 5: Quantitative comparison results on Mc-Master dataset with noise level $\sigma = 25$. The optimal results are indicated by **bold underlining**.

weight initialization is also an option. In practice, we found that when given enough training time, the performance of both approaches is quite similar.

### 4.1.2 Image deblurring

We validated the effectiveness of the proposed IR-CM model for image deblurring task based on the GoPro[43] dataset. A total of 2103 images are used for training, while 1111 images are reserved for testing. Our method is qualitatively and quantitatively compared with some of the milestone image deblurring methods including DeepDeblur[43], DeblurGAN-v2[14], DBGAN[15], MAXIM[4], DiffIR[44] and of course IR-SDE[22]. The comparison results are shown in Table 4 and Figure 4. More visual results are available in appendix D.

Our proposed method achieves optimal performance on SSIM and LPIPS metrics, surpassing the IR-SDE baseline model as well. This strongly demonstrates the effectiveness of our proposed consistency training approach. Notably, like other SDE-based models, we prioritize perceptual scores LPIPS over distortion scores PSNR and SSIM.

### 4.1.3 Image denoising

Note that the last term in (1) is a Gaussian process. Hence, we can consider a special case where $\mu = x(0)$. In this case, the IR-SDE degenerates into a pure additive noise process. This implies that any point along the PF-ODE trajectory can serve as a noisy image to be processed. In this scenario, we can only apply the conventional consistency training approach and cannot utilize the proposed linear-nonlinear decoupled training strategy because $x(0)$ must be set as the HQ image and $x(t)$ represents the corresponding low-quality (LQ) image.

Similar to [22], we collected approximately 5000 HQ images from the DIV2K[45], Flickr2K[45], and BSD500[46] datasets for training, and subsequently tested on the McMaster[47] dataset. To demonstrate the competitiveness of our method against state-of-the-art approaches, we compare it with DnCNN[7], FFDNet[8], as well as the special denoising methods proposed in [22], namely Denoising-SDE and Denoising-ODE. Our method achieves the optimal result in perceptual scores. Comparative experimental results are shown in Table 5, and visual results are available in appendix D.

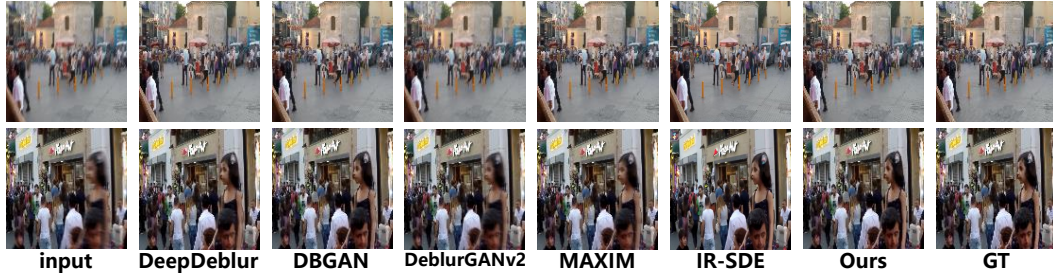

|  |  |  |  |  |  |  |  |
|---|---|---|---|---|---|---|---|
| input | DeepDeblur | DBGAN | DeblurGANv2 | MAXIM | IR-SDE | Ours | GT |

Figure 4: Qualitative comparison results on GoPro dataset. More visual results are available in appendix D.

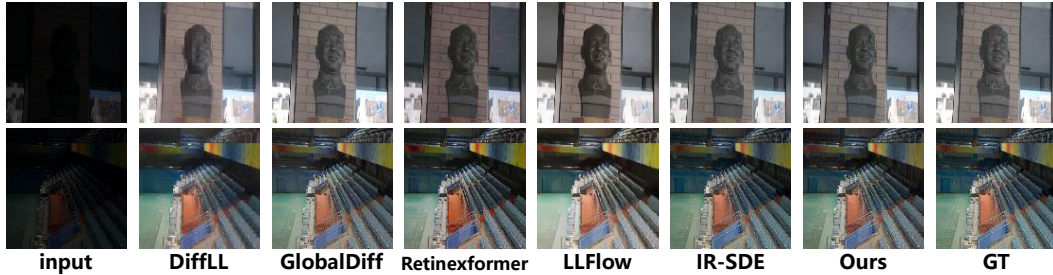

|  |  |  |  |  |  |  |  |
|---|---|---|---|---|---|---|---|
| input | DiffLL | GlobalDiff | Retinexformer | LLFlow | IR-SDE | Ours | GT |

Figure 5: Qualitative comparison results on LOLv2 dataset. More visual results are available in appendix D.

| Methods | Metrics | | | |
|---|---|---|---|---|
| Methods / Metrics | PSNR↑ | SSIM↑ | LPIPS↓ | NFE↓ |
| DiffLL[21] | 28.86 | 0.88 | 0.207 | 10 |
| GlobalDiff[20] | 28.82 | 0.90 | 0.095 | 500 |
| Retinexformer[16] | 27.71 | 0.86 | 0.097 | 1 |
| LLFlow[17] | 26.02 | 0.93 | 0.099 | 1 |
| IR-SDE[22] | 27.05 | 0.90 | 0.087 | 100 |
| CNN-baseline | 23.16 | 0.81 | 0.156 | 1 |
| IR-CM-CD (ours) | 26.33 | 0.89 | 0.098 | 1 |
| IR-CM-CT (ours) | 27.61 | 0.93 | 0.027 | 1 |
| IR-CM-CT (ours) | **30.20** | **0.95** | **0.021** | 2 |

Table 6: Quantitative comparison results on LOLv2 dataset. The optimal results are indicated by **bold underlining**.

| Group | Components | | | Metrics | | |
|---|---|---|---|---|---|---|
|  | OECF | OGL | LLDT | PSNR | SSIM | LPIPS |
| 1 | × | × | × | 23.06 | 0.82 | 0.144 |
| 2 | × | ✓ | × | 23.62 | 0.88 | 0.094 |
| 3 | ✓ | × | × | 23.99 | 0.86 | 0.120 |
| 4 | ✓ | ✓ | × | 25.04 | 0.91 | 0.059 |
| 5 | × | × | ✓ | 23.96 | 0.84 | 0.121 |
| 6 | × | ✓ | ✓ | 25.01 | 0.89 | 0.072 |
| 7 | ✓ | × | ✓ | 24.45 | 0.86 | 0.101 |
| 8 | ✓ | ✓ | ✓ | 27.61 | 0.93 | 0.027 |

Table 7: Ablation experiment results on LOLv2 dataset. OECF: origin-estimated consistency function, OGL: origin-guided loss, LLDT: linear-nonlinear decoupling training strategy.

#### 4.1.4 Low-light image enhancement

We compare our method with existing researches for low-light image enhancement tasks on the LOLv2[48] dataset consisted of 689 pairs training images and 100 pairs testing images. We compare our proposed method with some recent approaches for low-light image enhancement task, including: DiffLL[21], GlobalDiff[20], Retinexformer[16], LLFlow[17]. Note that IR-SDE is trained following the experimental settings described in [22], while IR-CM-CD is based on consistency distillation using the trained weights of IR-SDE. Our method outperforms the state-of-the-art diffusion-based methods, DiffLL and GlobalDiff, and requires only two sampling steps. The corresponding quantitative and qualitative comparison results are shown in Table 6 and Figure 5 respectively. More visual results are available in appendix D

#### 4.1.5 Runtime Comparation

To validate the superiority of our method in inference real-time performance, we compared it with several other state-of-the-art (SOTA) methods across three common resolution sizes. The results are shown in Table 8. Compared to the baseline method IR-SDE, our method significantly reduces inference time by introducing a consistency training process. In some real-time demanding applications, one-step sampling inference allows for fast predictions while maintaining competitive performance. On the other hand, two-step sampling inference can substantially improve model

performance with only a slight trade-off in inference speed. Users can choose the approach based on their specific requirements. Additionally, a discussion on the model complexity will be provided in Appendix E.

| image size / methods | MAXIM | Restormer | IR-SDE | DiffLL | GlobalDiff | DiffIR | IR-CM(1-step) | IR-CM(2-step) |
|---|---|---|---|---|---|---|---|---|
| 256x256 | 0.092 | 0.117 | 7.325 | 0.087 | 0.134 | 0.574 | **_0.073_** | 0.145 |
| 600x400 | 0.297 | 0.395 | 27.418 | 0.309 | 0.637 | 2.743 | **_0.273_** | 0.547 |
| 1280x720 | 1.223 | 1.512 | 98.731 | 1.045 | 1.881 | 7.155 | **_0.992_** | 1.984 |

Table 8: Inference time comparison with some of SOTA methods on three typical image sizes. All tests were conducted using an NVIDIA 2080Ti GPU. The optimal results are indicated by **bold underlining**.

## 4.2 Ablation Experiments

### 4.2.1 Components ablation experiments

The main innovations of this paper lie in the OECF, Origin-guided loss, and the linear-nonlinear decoupling training strategy proposed in the previous section. To verify the effectiveness of each component, we conducted a series of ablation experiments using low-light image enhancement as an example. To ensure fairness, each control group used the same checkpoint for model initialization, and the total number of epochs was kept consistent during training. And only the performance of one-step sampling inference was considered. The corresponding results are shown in Table 7. The effectiveness of the proposed OECF can be demonstrated by group pairs (1, 3), (2, 4), (5, 7), (6, 8). The significant improvement in perceptual scores due to the origin-guided loss is evident from the comparisons between group pairs (1, 2), (3, 4), (5, 6), (7, 8). Finally, the notable performance enhancement of the model due to the linear-nonlinear decoupling training strategy can be seen from the comparisons between groups (1, 5), (2, 6), (3, 7), (4, 8). Note that for groups 5 and 7, we only use the origin-guided loss during the Linear-fitting stage, and not during the nonlinear-fitting stage.

### 4.2.2 Selection of origin-guided loss weight

The setting of $\lambda_{OG}$ also affects the performance of IR-CM. If set too low, the model may become unstable and perform poorly, as discussed in Section 3.2. If set too high, it will affect self-consistency property, leading to a degradation into a CNN-baseline method. We experimented with multiple values and empirically found that the best performance is achieved around 0.8. The related experimental results are discussed in Appendix B.

### 4.2.3 Linear-fitting stage only & Multiple sampling

In fact, after completing the linear-fitting stage, the model can already perform one-step sampling inference, however, it cannot perform multi-step sampling as discussed in Section 3.3. To further evaluate the model's performance, we tested it under the conditions of linear-fitting stage only, and with 1-step, 2-step, 4-step, and 6-step sampling. Empirically, we consider 2-step sampling to be the most cost-effective choice. The related experimental results are discussed in Appendix C.

## 5 Conclusion

This paper proposes a multi-task image restoration and enhancement method, IR-CM, based on a consistency training approach, enabling few-step or even one-step sampling inference. Specifically, we proposed the Origin-estimated Consistency Function (OECF), which provides a more stable initial state and a smaller solution space for the consistency training process. Furthermore, to make the training process more robust and prevent trivial solutions, we introduced the Origin-guided Loss (OE Loss). Based on these, we developed a Linear-Nonlinear Decoupling Training Strategy, which not only accelerates the training process but, more importantly, enables the model to perform multi-step sampling inference, further enhancing its performance. Finally, a series of comparative experiments and ablation studies demonstrated the effectiveness of the proposed method.

# 6 Acknowledgment

Our work is supported by the weapons and equipment pre-research fund, Grant No. 50911020603.

# References

[1] Dongwei Ren, Wangmeng Zuo, Qinghua Hu, Pengfei Zhu, and Deyu Meng. Progressive image deraining networks: A better and simpler baseline. In *Proceedings of the IEEE/CVF conference on computer vision and pattern recognition*, pages 3937–3946, 2019.

[2] Syed Waqas Zamir, Aditya Arora, Salman Khan, Munawar Hayat, Fahad Shahbaz Khan, Ming-Hsuan Yang, and Ling Shao. Multi-stage progressive image restoration. In *Proceedings of the IEEE/CVF conference on computer vision and pattern recognition*, pages 14821–14831, 2021.

[3] Wenhan Yang, Robby T Tan, Jiashi Feng, Zongming Guo, Shuicheng Yan, and Jiaying Liu. Joint rain detection and removal from a single image with contextualized deep networks. *IEEE transactions on pattern analysis and machine intelligence*, 42(6):1377–1393, 2019.

[4] Zhengzhong Tu, Hossein Talebi, Han Zhang, Feng Yang, Peyman Milanfar, Alan Bovik, and Yinxiao Li. Maxim: Multi-axis mlp for image processing. In *Proceedings of the IEEE/CVF conference on computer vision and pattern recognition*, pages 5769–5780, 2022.

[5] Syed Waqas Zamir, Aditya Arora, Salman Khan, Munawar Hayat, Fahad Shahbaz Khan, and Ming-Hsuan Yang. Restormer: Efficient transformer for high-resolution image restoration. In *Proceedings of the IEEE/CVF conference on computer vision and pattern recognition*, pages 5728–5739, 2022.

[6] Yeachan Park, Myeongho Jeon, Junho Lee, and Myungjoo Kang. Mcw-net: Single image deraining with multi-level connections and wide regional non-local blocks. *Signal Processing: Image Communication*, 105:116701, 2022.

[7] Kai Zhang, Wangmeng Zuo, Yunjin Chen, Deyu Meng, and Lei Zhang. Beyond a gaussian denoiser: Residual learning of deep cnn for image denoising. *IEEE transactions on image processing*, 26(7):3142–3155, 2017.

[8] Kai Zhang, Wangmeng Zuo, and Lei Zhang. Ffdnet: Toward a fast and flexible solution for cnn-based image denoising. *IEEE Transactions on Image Processing*, 27(9):4608–4622, 2018.

[9] Rongkai Zhang, Jiang Zhu, Zhiyuan Zha, Justin Dauwels, and Bihan Wen. R3l: Connecting deep reinforcement learning to recurrent neural networks for image denoising via residual recovery. In *2021 IEEE International Conference on Image Processing (ICIP)*, pages 1624–1628. IEEE, 2021.

[10] Shen Cheng, Yuzhi Wang, Haibin Huang, Donghao Liu, Haoqiang Fan, and Shuaicheng Liu. Nbnet: Noise basis learning for image denoising with subspace projection. In *Proceedings of the IEEE/CVF conference on computer vision and pattern recognition*, pages 4896–4906, 2021.

[11] Meng Chang, Qi Li, Huajun Feng, and Zhihai Xu. Spatial-adaptive network for single image denoising. In *Computer Vision–ECCV 2020: 16th European Conference, Glasgow, UK, August 23–28, 2020, Proceedings, Part XXX 16*, pages 171–187. Springer, 2020.

[12] Seungjun Nah, Tae Hyun Kim, and Kyoung Mu Lee. Deep multi-scale convolutional neural network for dynamic scene deblurring. In *Proceedings of the IEEE conference on computer vision and pattern recognition*, pages 3883–3891, 2017.

[13] Orest Kupyn, Volodymyr Budzan, Mykola Mykhailych, Dmytro Mishkin, and Jiří Matas. Deblurgan: Blind motion deblurring using conditional adversarial networks. In *Proceedings of the IEEE conference on computer vision and pattern recognition*, pages 8183–8192, 2018.

[14] Orest Kupyn, Tetiana Martyniuk, Junru Wu, and Zhangyang Wang. Deblurgan-v2: Deblurring (orders-of-magnitude) faster and better. In *Proceedings of the IEEE/CVF international conference on computer vision*, pages 8878–8887, 2019.

[15] Kaihao Zhang, Wenhan Luo, Yiran Zhong, Lin Ma, Bjorn Stenger, Wei Liu, and Hongdong Li. Deblurring by realistic blurring. In *Proceedings of the IEEE/CVF conference on computer vision and pattern recognition*, pages 2737–2746, 2020.

[16] Yuanhao Cai, Hao Bian, Jing Lin, Haoqian Wang, Radu Timofte, and Yulun Zhang. Retinexformer: One-stage retinex-based transformer for low-light image enhancement. In *Proceedings of the IEEE/CVF International Conference on Computer Vision (ICCV)*, pages 12504–12513, October 2023.

[17] Yufei Wang, Renjie Wan, Wenhan Yang, Haoliang Li, Lap-Pui Chau, and Alex C Kot. Low-light image enhancement with normalizing flow. *arXiv preprint arXiv:2109.05923*, 2021.

[18] Alexandru Brateanu, Raul Balmez, Adrian Avram, and Ciprian Orhei. Lyt-net: Lightweight yuv transformer-based network for low-light image enhancement. *arXiv preprint arXiv:2401.15204*, 2024.

[19] Wu Yuhui, Pan Chen, Wang Guoqing, Yang Yang, Wei Jiwei, Li Chongyi, and Heng Tao Shen. Learning semantic-aware knowledge guidance for low-light image enhancement. In *Proceedings of the IEEE/CVF Conference on Computer Vision and Pattern Recognition*, 2023.

[20] Hou Jinhui, Zhu Zhiyu, Liu Hui, Zeng Huanqiang, and Yuan Hui. Global structure-aware diffusion process for low-light image enhancement. *Advances in Neural Information Processing Systems*, 2023.

[21] Hai Jiang, Ao Luo, Haoqiang Fan, Songchen Han, and Shuaicheng Liu. Low-light image enhancement with wavelet-based diffusion models. *ACM Transactions on Graphics (TOG)*, 42(6):1–14, 2023.

[22] Ziwei Luo, Fredrik K Gustafsson, Zheng Zhao, Jens Sjölund, and Thomas B Schön. Image restoration with mean-reverting stochastic differential equations. *International Conference on Machine Learning*, 2023.

[23] Prafulla Dhariwal and Alexander Nichol. Diffusion models beat gans on image synthesis. *Advances in neural information processing systems*, 34:8780–8794, 2021.

[24] Jonathan Ho, Ajay Jain, and Pieter Abbeel. Denoising diffusion probabilistic models. *Advances in neural information processing systems*, 33:6840–6851, 2020.

[25] Jiaming Song, Chenlin Meng, and Stefano Ermon. Denoising diffusion implicit models. *arXiv:2010.02502*, October 2020.

[26] Alexia Jolicoeur-Martineau, Ke Li, Rémi Piché-Taillefer, Tal Kachman, and Ioannis Mitliagkas. Gotta go fast when generating data with score-based models. *arXiv preprint arXiv:2105.14080*, 2021.

[27] Chitwan Saharia, Jonathan Ho, William Chan, Tim Salimans, David J Fleet, and Mohammad Norouzi. Image super-resolution via iterative refinement. *IEEE transactions on pattern analysis and machine intelligence*, 45(4):4713–4726, 2022.

[28] Andreas Lugmayr, Martin Danelljan, Andres Romero, Fisher Yu, Radu Timofte, and Luc Van Gool. Repaint: Inpainting using denoising diffusion probabilistic models. In *Proceedings of the IEEE/CVF conference on computer vision and pattern recognition*, pages 11461–11471, 2022.

[29] Jonathan Ho, Chitwan Saharia, William Chan, David J Fleet, Mohammad Norouzi, and Tim Salimans. Cascaded diffusion models for high fidelity image generation. *Journal of Machine Learning Research*, 23(47):1–33, 2022.

[30] Ziwei Luo, Fredrik K Gustafsson, Zheng Zhao, Jens Sjölund, and Thomas B Schön. Refusion: Enabling large-size realistic image restoration with latent-space diffusion models. In *Proceedings of the IEEE/CVF Conference on Computer Vision and Pattern Recognition Workshops*, pages 1680–1691, 2023.

[31] Ziwei Luo, Fredrik K Gustafsson, Zheng Zhao, Jens Sjölund, and Thomas B Schön. Controlling vision-language models for universal image restoration. *arXiv preprint arXiv:2310.01018*, 2023.

[32] Ziwei Luo, Fredrik K Gustafsson, Zheng Zhao, Jens Sjölund, and Thomas B Schön. Photorealistic image restoration in the wild with controlled vision-language models. *arXiv preprint arXiv:2404.09732*, 2024.

[33] Yang Song and Stefano Ermon. Generative modeling by estimating gradients of the data distribution. *Advances in neural information processing systems*, 32, 2019.

[34] Yang Song, Jascha Sohl-Dickstein, Diederik P Kingma, Abhishek Kumar, Stefano Ermon, and Ben Poole. Score-based generative modeling through stochastic differential equations. *arXiv preprint arXiv:2011.13456*, 2020.

[35] Cheng Lu, Yuhao Zhou, Fan Bao, Jianfei Chen, Chongxuan Li, and Jun Zhu. Dpm-solver: A fast ode solver for diffusion probabilistic model sampling in around 10 steps. *Advances in Neural Information Processing Systems*, 35:5775–5787, 2022.

[36] Kaiwen Zheng, Cheng Lu, Jianfei Chen, and Jun Zhu. Dpm-solver-v3: Improved diffusion ode solver with empirical model statistics. *Advances in Neural Information Processing Systems*, 36, 2024.

[37] Yang Song, Prafulla Dhariwal, Mark Chen, and Ilya Sutskever. Consistency models. *arXiv preprint arXiv:2303.01469*, 2023.

[38] Karen Simonyan and Andrew Zisserman. Very deep convolutional networks for large-scale image recognition. *arXiv preprint arXiv:1409.1556*, 2014.

[39] Zhou Wang, Alan C Bovik, Hamid R Sheikh, and Eero P Simoncelli. Image quality assessment: from error visibility to structural similarity. *IEEE transactions on image processing*, 13(4):600–612, 2004.

[40] Richard Zhang, Phillip Isola, Alexei A Efros, Eli Shechtman, and Oliver Wang. The unreasonable effectiveness of deep features as a perceptual metric. In *Proceedings of the IEEE conference on computer vision and pattern recognition*, pages 586–595, 2018.

[41] Wenhan Yang, Robby T Tan, Jiashi Feng, Jiaying Liu, Zongming Guo, and Shuicheng Yan. Deep joint rain detection and removal from a single image. In *Proceedings of the IEEE conference on computer vision and pattern recognition*, pages 1357–1366, 2017.

[42] Rui Qian, Robby T Tan, Wenhan Yang, Jiajun Su, and Jiaying Liu. Attentive generative adversarial network for raindrop removal from a single image. In *Proceedings of the IEEE conference on computer vision and pattern recognition*, pages 2482–2491, 2018.

[43] Tae Hyun Kim, Seungjun Nah, and Kyoung Mu Lee. Deep multi-scale convolutional neural network for dynamic scene deblurring. In *Conference on Computer Vision and Pattern Recognition*, pages 1–21. IEEE, 2017.

[44] Bin Xia, Yulun Zhang, Shiyin Wang, Yitong Wang, Xinglong Wu, Yapeng Tian, Wenming Yang, and Luc Van Gool. Diffir: Efficient diffusion model for image restoration. In *Proceedings of the IEEE/CVF International Conference on Computer Vision*, pages 13095–13105, 2023.

[45] Eirikur Agustsson and Radu Timofte. Ntire 2017 challenge on single image super-resolution: Dataset and study. In *Proceedings of the IEEE conference on computer vision and pattern recognition workshops*, pages 126–135, 2017.

[46] Pablo Arbelaez, Michael Maire, Charless Fowlkes, and Jitendra Malik. Contour detection and hierarchical image segmentation. *IEEE transactions on pattern analysis and machine intelligence*, 33(5):898–916, 2010.

[47] Lei Zhang, Xiaolin Wu, Antoni Buades, and Xin Li. Color demosaicking by local directional interpolation and nonlocal adaptive thresholding. *Journal of Electronic imaging*, 20(2):023016–023016, 2011.

[48] Wenhan Yang, Wenjing Wang, Haofeng Huang, Shiqi Wang, and Jiaying Liu. Sparse gradient regularized deep retinex network for robust low-light image enhancement. *IEEE Transactions on Image Processing*, 30:2072–2086, 2021.

[49] Ilya Loshchilov and Frank Hutter. Decoupled weight decay regularization. *arXiv preprint arXiv:1711.05101*, 2017.

# A Implementation details

We chose the same U-Net backbone network as used in IR-SDE[22]. Note that the CNN-baseline uses the same U-Net backbone, with its time embedding set to a constant, and it directly outputs HQ images from the input LQ images. For all tasks, we set the training patch-size to be $128 \times 128$ and use a batch size of 4. We used the AdamW[49] optimizer with a weight decay set to 0.01 and set learning rate to $10^{-5}$. Our models are trained on two 2080Ti GPUs for 200 epochs for each task, with the linear-fitting stage and the nonlinear-fitting stage each accounting for 100 epochs. The training hyperparameters are set as follows: $\lambda = 10, \lambda_{OG} = 0.8, \lambda_{perc} = 0.125$.

The $\theta$ schedule is defined same as [22]:

$$\theta_t = 1 - \frac{f(t)}{f(0)}, \quad f(t) = \cos(\frac{t/T + s}{1 + s} \cdot \frac{\pi}{2})^2, \tag{17}$$

where $s = 0.008$. And the consistency weights $c_{skip}$ and $c_{out}$ are set as follow:

$$c_{skip}(t) = 1 - c_{out}(t), \quad c_{out}(t) = t/T, \tag{18}$$

where $T$ is set to 100.

# B Selection of origin-guided loss weight

We take the low-light image enhancement task as an example to consider the impact of different $\lambda_{OG}$ values on the final performance of the model considering only the one-step sampling scenario. The quantitative experimental results are shown in Table 9.

| Metrics/$\lambda_{OG}$ | 0 | 0.2 | 0.4 | 0.6 | 0.8 | 1.0 | 1.2 |
|---|---|---|---|---|---|---|---|
| PSNR | 22.36 | 22.11 | 24.42 | 26.94 | 27.61 | **27.67** | 27.53 |
| SSIM | 0.85 | 0.87 | 0.89 | 0.92 | **0.93** | 0.92 | 0.90 |
| LPIPS | 0.136 | 0.129 | 0.086 | 0.044 | **0.027** | 0.042 | 0.075 |

Table 9: Quantitative comparison results of different $\lambda_{OG}$. The optimal results are indicated by **bold underlining**.

The results show that the best performance is achieved when $\lambda_{OG}$ is set to 0.8. Although a higher PSNR score was obtained with $\lambda_{OG} = 1.0$, we prioritize perceptual scores. Therefore, we chose $\lambda_{OG} = 0.8$ as the experimental setting in Section 4.1.

# C linear-fitting stage only & Multiple sampling

As shown in Algorithm 3, our method, like most SDE-based methods, can improve model performance through multi-step sampling. Specifically, if set $M = 1$, one-step sampling can be achieved. To further evaluate the model's performance, we tested it under 1-step, 2-step, 3-step, and 4-step sampling. Additionally, we evaluated the model's performance after completing only the linear-fitting stage (LFS). Similarly, using low-light enhancement as an example, the test results are shown in Table 10 and Figure 6.

| Metrics/NFE | 1 | 2 | 3 | 4 | LFS only (one-step) |
|---|---|---|---|---|---|
| PSNR | 27.6139 | 30.2070 | **31.4882** | 31.2916 | 27.2421 |
| SSIM | 0.9301 | 0.9493 | **0.9496** | 0.9490 | 0.9216 |
| LPIPS | 0.02693 | **0.02080** | 0.02146 | 0.02247 | 0.02803 |

Table 10: Quantitative comparison results of different sampling step. The optimal results are indicated by **bold underlining**.

It can be observed that the optimal perceptual score is achieved when the sampling step is 2. Further increasing the sampling steps leads to only slight improvements in model performance but implies longer inference times. In the case of one-step sampling, the performance of LFS only is not significantly different from that of the two-stage trained model. Considering all factors, we believe that 2-step sampling inference is the most cost-effective.

**Algorithm 3:** Multi-step sampling inference

**Input** LQ image $x_{LQ}$, OECF $f_\phi(\cdot, \cdot)$, sequence of time points $t_1 < t_2 ... < t_N$, inference step $M$;
$m \leftarrow 0, n \leftarrow N, \mu \leftarrow x_{LQ}$;
Sample $x_{t_n} \sim \mathcal{N}(\mu, v_{t_n})$;
**while** $m < M$ **do**
$\quad x_0 \leftarrow f_\phi(x_{t_n}, t_n)$;
$\quad n \leftarrow n/2$;
$\quad m_{t_n} \leftarrow \mu + (x_0 - \mu)e^{-\bar{\theta}_{t_n}}$;
$\quad$ Sample $x_{t_n} \sim \mathcal{N}(m_{t_n}, v_{t_n})$;
$\quad m \leftarrow m + 1$
**end**
**Output:** $x_0$

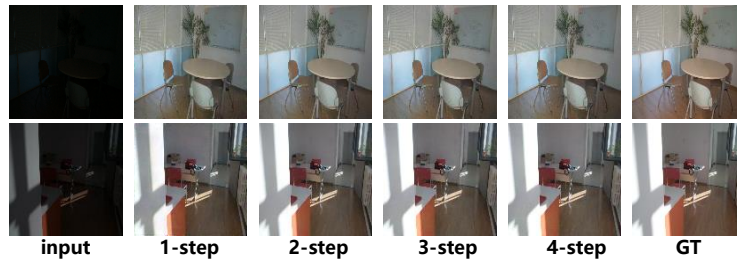

| input | 1-step | 2-step | 3-step | 4-step | GT |

Figure 6: Visual results of different sampling step.

## D  More visual results

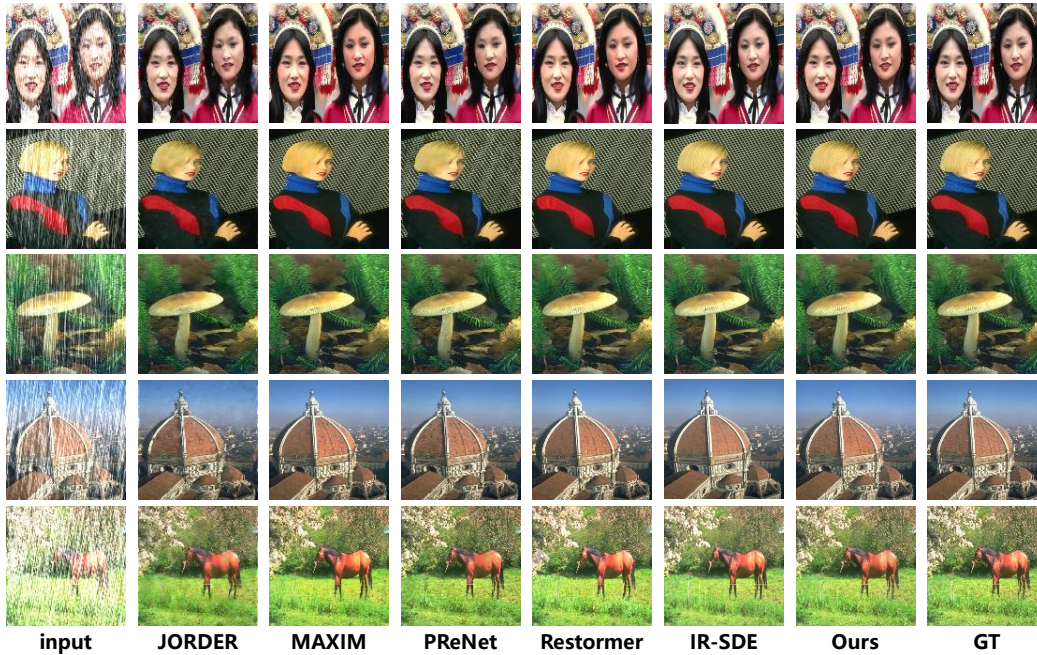

| input | JORDER | MAXIM | PReNet | Restormer | IR-SDE | Ours | GT |

Figure 7: Image deraining visual results on R100H dataset.

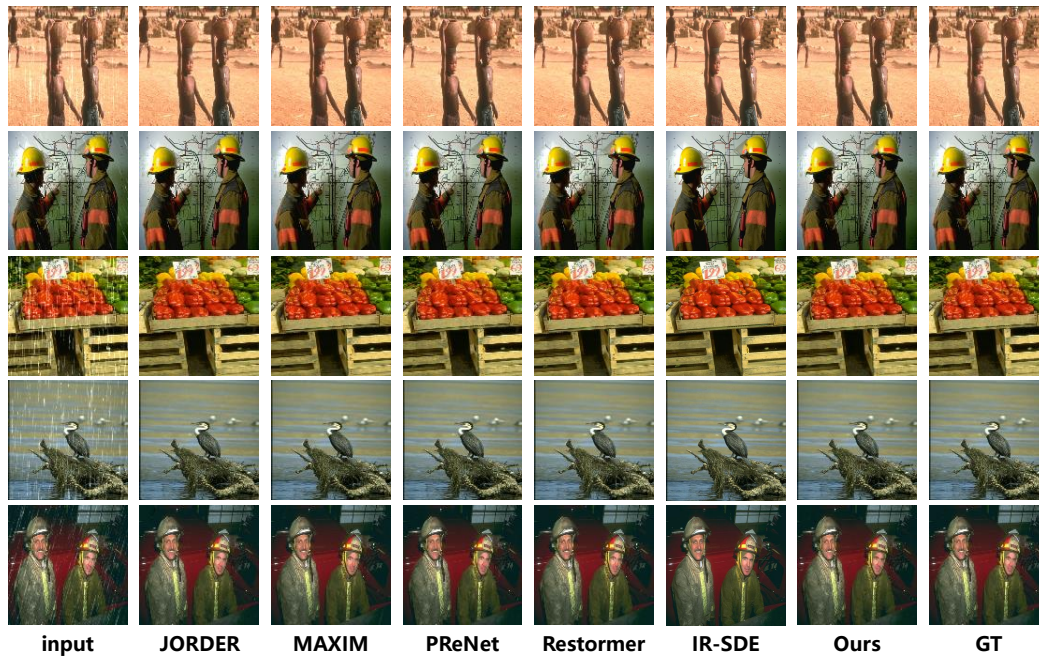

Figure 8: Image deraining visual results on R100L dataset.

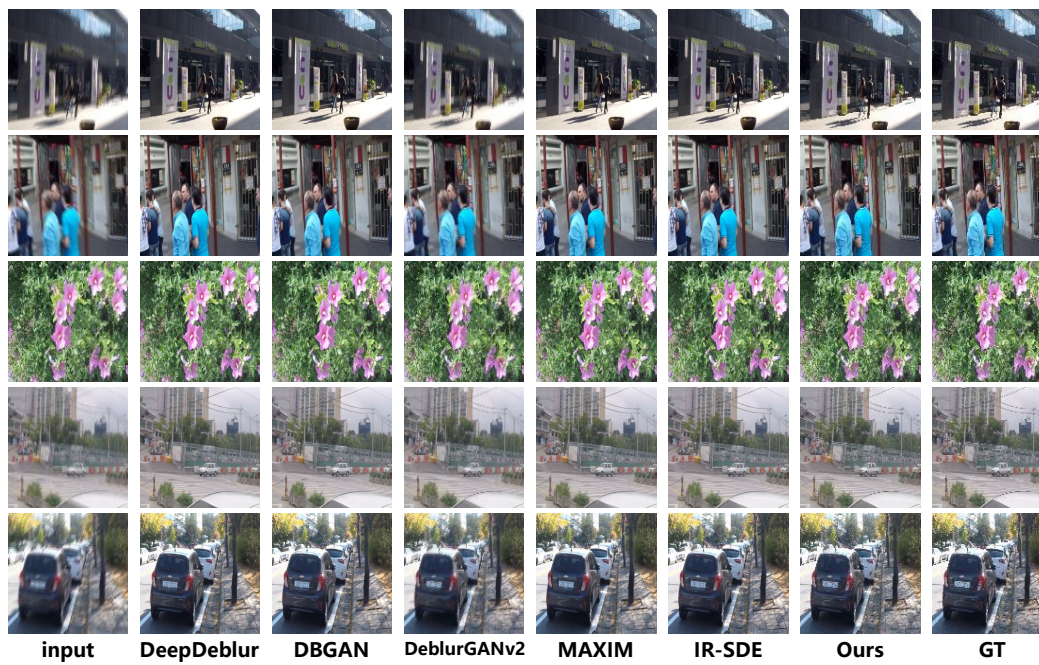

Figure 9: Image deblurring visual results on GoPro dataset.

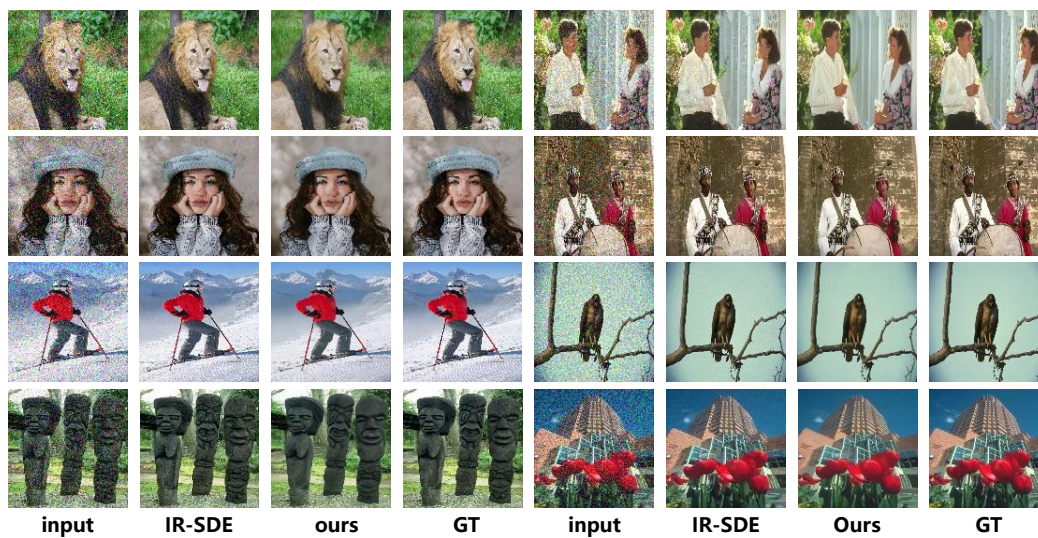

Figure 10: Image denoising visual results on McMaster dataset with $sigma = 25$.

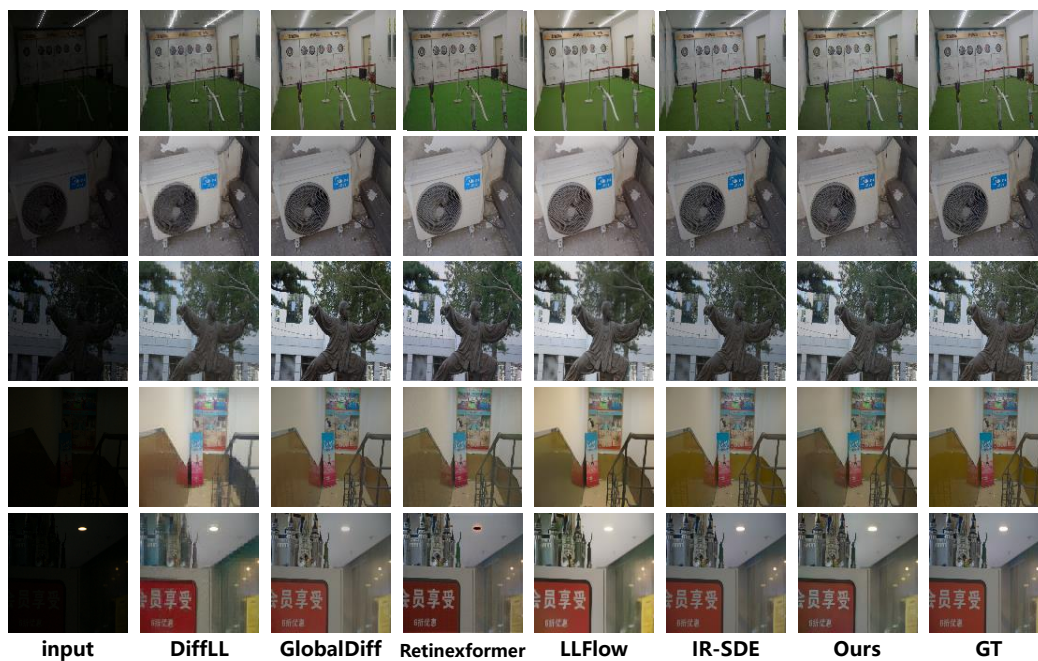

Figure 11: Low-light image enhancement visual results on LOLv2 dataset.

# E Model complexity

Our backbone network is a conditional Unet, and its computational complexity is proportional to the number of convolutional layers and the size of the image. The complexity can be calculated as follows:

$$O(S * \sum_{i=0}^{L-1} (\frac{H}{2i} * \frac{W}{2i} * C_{in,i} * C_{out,i} * K^2)), \tag{19}$$

where $L$ is number of convolutional layers, $H, W$ are the height and width of the image, $C_{in,i}, C_{out,i}$ represent the input and output sizes of the convolutional layer respectively, $K$ is the size of the convolutional kernel, and $S$ denotes the sample steps. From the above formula, it can be seen that our model has a linear relationship with the height and width of the input image. And it is also linearly related to the number of sampling steps. This is advantageous for applying our model to larger-sized images. Additionally, by adjusting the number of sampling steps, a trade-off between real-time performance and model performance can be achieved.

# F Potential societal impact

Our work on conditional image generation tasks, such as image restoration and enhancement, has several potential societal impacts. These improvements can benefit fields like medical imaging, where enhanced image quality can aid in better diagnosis and treatment. Additionally, improved image restoration techniques can be valuable in preserving and restoring historical photographs and artworks.

However, there are also potential negative impacts to consider. Enhanced image generation techniques could be misused for creating deceptive content, such as deepfakes, which can have serious ethical and social implications. Therefore, it is crucial to implement safeguards and ethical guidelines to prevent misuse and ensure that the technology is used for beneficial purposes.

We are committed to promoting the positive applications of our research while being aware of and mitigating potential risks.

